# Prediction and Change Detection

**Mark Steyvers**
*msteyver@uci.edu*
University of California, Irvine
Irvine, CA 92697

**Scott Brown**
*scottb@uci.edu*
University of California, Irvine
Irvine, CA 92697

## Abstract

We measure the ability of human observers to predict the next datum in a sequence that is generated by a simple statistical process undergoing change at random points in time. Accurate performance in this task requires the identification of changepoints. We assess individual differences between observers both empirically, and using two kinds of models: a Bayesian approach for change detection and a family of cognitively plausible fast and frugal models. Some individuals detect too many changes and hence perform sub-optimally due to excess variability. Other individuals do not detect enough changes, and perform sub-optimally because they fail to notice short-term temporal trends.

## 1 Introduction

Decision-making often requires a rapid response to change. For example, stock analysts need to quickly detect changes in the market in order to adjust investment strategies. Coaches need to track changes in a player's performance in order to adjust strategy. When tracking changes, there are costs involved when either more or less changes are observed than actually occurred. For example, when using an overly conservative change detection criterion, a stock analyst might miss important short-term trends and interpret them as random fluctuations instead. On the other hand, a change may also be detected too readily. For example, in basketball, a player who makes a series of consecutive baskets is often identified as a "hot hand" player whose underlying ability is perceived to have suddenly increased [1,2]. This might lead to sub-optimal passing strategies, based on random fluctuations.

We are interested in explaining individual differences in a sequential prediction task. Observers are shown stimuli generated from a simple statistical process with the task of predicting the next datum in the sequence. The latent parameters of the statistical process change discretely at random points in time. Performance in this task depends on the accurate detection of those changepoints, as well as inference about future outcomes based on the outcomes that followed the most recent inferred changepoint. There is much prior research in statistics on the problem of identifying changepoints [3,4,5]. In this paper, we adopt a Bayesian approach to the changepoint identification problem and develop a simple inference procedure to predict the next datum in a sequence. The Bayesian model serves as an ideal observer model and is useful to characterize the ways in which individuals deviate from optimality.

The plan of the paper is as follows. We first introduce the sequential prediction task and discuss a Bayesian analysis of this prediction problem. We then discuss the results from a few individuals in this prediction task and show how the Bayesian approach can capture individual differences with a single "twitchiness" parameter that describes how readily changes are perceived in random sequences. We will show that some individuals are too twitchy: their performance is too variable because they base their predictions on too little of the recent data. Other individuals are not twitchy enough, and they fail to capture fast changes in the data. We also show how behavior can be explained with a set of *fast and frugal* models [6]. These are cognitively realistic models that operate under plausible computational constraints.

## 2   A prediction task with multiple changepoints

In the prediction task, stimuli are presented sequentially and the task is to predict the next stimulus in the sequence. After $t$ trials, the observer has been presented with stimuli $y_1, y_2, \ldots, y_t$ and the task is to make a prediction about $y_{t+1}$. After the prediction is made, the actual outcome $y_{t+1}$ is revealed and the next trial proceeds to the prediction of $y_{t+2}$. This procedure starts with $y_1$ and is repeated for $T$ trials.

The observations $y_t$ are $D$-dimensional vectors with elements sampled from binomial distributions. The parameters of those distributions change discretely at random points in time such that the mean increases or decreases after a change point. This generates a sequence of observation vectors, $y_1, y_2, \ldots, y_T$, where each $y_t = \{y_{t,1} \ldots y_{t,D}\}$. Each of the $y_{t,d}$ is sampled from a binomial distribution $\text{Bin}(\theta_{t,d}, K)$, so $0 \leq y_{t,d} \leq K$. The parameter vector $\theta_t = \{\theta_{t,1} \ldots \theta_{t,D}\}$ changes depending on the locations of the changepoints. At each time step, $x_t$ is a binary indicator for the occurrence of a changepoint occurring at time $t+1$. The parameter $\alpha$ determines the probability of a change occurring in the sequence. The generative model is specified by the following algorithm:

1. For $d=1..D$ sample $\theta_{1,d}$ from a Uniform(0,1) distribution

2. For $t=2..T$,

    (a) Sample $x_{t-1}$ from a Bernoulli($\alpha$) distribution

    (b) If $x_{t-1}=0$, then $\theta_t=\theta_{t-1}$, else

        for $d=1..D$ sample $\theta_{t,d}$ from a Uniform(0,1) distribution

    (c) for $d=1..D$, sample $y_t$ from a $\text{Bin}(\theta_{t,d}, K)$ distribution

Table 1 shows some data generated from the changepoint model with $T=20$, $\alpha=.1$, and $D=1$. In the prediction task, $y$ will be observed, but $x$ and $\theta$ are not.

Table 1: Example data

| $t$ | 1 | 2 | 3 | 4 | 5 | 6 | 7 | 8 | 9 | 10 | 11 | 12 | 13 | 14 | 15 | 16 | 17 | 18 | 19 | 20 |
|---|---|---|---|---|---|---|---|---|---|---|---|---|---|---|---|---|---|---|---|---|
| $x$ | 0 | 0 | 0 | 1 | 0 | 0 | 1 | 0 | 0 | 0 | 0 | 0 | 0 | 1 | 0 | 1 | 0 | 0 | 0 | 0 |
| $\theta$ | .68 | .68 | .68 | .68 | .48 | .48 | .48 | .74 | .74 | .74 | .74 | .74 | .74 | .19 | .19 | .87 | .87 | .87 | .87 | .87 |
| $y$ | 9 | 7 | 8 | 7 | 4 | 4 | 4 | 9 | 8 | 3 | 6 | 7 | 8 | 2 | 1 | 8 | 9 | 9 | 8 | 8 |

# 3   A Bayesian prediction model

In both our Bayesian and fast-and-frugal analyses, the prediction task is decomposed into two inference procedures. First, the changepoint locations are identified. This is followed by predictive inference for the next outcome based on the most recent changepoint locations. Several Bayesian approaches have been developed for changepoint problems involving single or multiple changepoints [3,5]. We apply a Markov Chain Monte Carlo (MCMC) analysis to approximate the joint posterior distribution over changepoint assignments $x$ while integrating out $\theta$. Gibbs sampling will be used to sample from this posterior marginal distribution. The samples can then be used to predict the next outcome in the sequence.

## 3.1   Inference for changepoint assignments.

To apply Gibbs sampling, we evaluate the conditional probability of assigning a changepoint at time $i$, given all other changepoint assignments and the current $\alpha$ value. By integrating out $\theta$, the conditional probability is

$$P\left(x_i \mid x_{-i}, y, \alpha\right) = \int_{\theta} P\left(x_i, \theta, \alpha \mid x_{-i}, y\right) \tag{1}$$

where $x_{-i}$ represents all switch point assignments except $x_i$. This can be simplified by considering the location of the most recent changepoint preceding and following time $i$ and the outcomes occurring between these locations. Let $n_i^L$ be the number of time steps from the last changepoint up to and including the current time step $i$ such that $x_{i-n_i^L} = 1$ and $x_{i-n_i^L+j} = 0$ for $0 < j < n_i^L$. Similarly, let $n_i^R$ be the number of time steps that follow time step $i$ up to the next changepoint such that $x_{i+n_i^R} = 1$ and $x_{i+n_i^R-j} = 0$ for $0 < j < n_i^R$. Let $y_i^L = \sum_{i-n_i^L < k \leq i} y_k$ and $y_i^R = \sum_{k < k \leq i+n_i^R} y_k$. The update equation for the changepoint assignment can then be simplified to

$$P\left(x_i = m \mid x_{-i}\right) \propto$$

$$\begin{cases} (1-\alpha)\prod_{j=1}^{D} \dfrac{\Gamma\left(1+y_{i,j}^L+y_{i,j}^R\right)\Gamma\left(1+Kn_i^L+Kn_i^R-y_{i,j}^L-y_{i,j}^R\right)}{\Gamma\left(2+Kn_i^L+Kn_i^R\right)} & m=0 \\[4mm] \alpha\prod_{j=1}^{D} \dfrac{\Gamma\left(1+y_{i,j}^L\right)\Gamma\left(1+Kn_i^L-y_{i,j}^L\right)\Gamma\left(1+y_{i,j}^R\right)\Gamma\left(1+Kn_i^R-y_{i,j}^R\right)}{\Gamma\left(2+Kn_i^L\right)\Gamma\left(2+Kn_i^R\right)} & m=1 \end{cases} \tag{2}$$

We initialize the Gibbs sampler by sampling each $x_t$ from a Bernoulli($\alpha$) distribution. All changepoint assignments are then updated sequentially by the Gibbs sampling equation above. The sampler is run for $M$ iterations after which one set of changepoint assignments is saved. The Gibbs sampler is then restarted multiple times until $S$ samples have been collected.

Although we could have included an update equation for $\alpha$, in this analysis we treat $\alpha$ as a known constant. This will be useful when characterizing the differences between human observers in terms of differences in $\alpha$.

## 3.2    Predictive inference

The next latent parameter value $\theta_{t+1}$ and outcome $y_{t+1}$ can be predicted on the basis of observed outcomes that occurred after the last inferred changepoint:

$$\theta_{t+1,j} = \sum_{i=t^*+1}^{t} y_{i,j} / K, \qquad y_{t+1,j} = \text{round}\left(\theta_{t+1,j}K\right) \tag{3}$$

where $t^*$ is the location of the most recent change point. By considering multiple Gibbs samples, we get a distribution over outcomes $y_{t+1}$. We base the model predictions on the mean of this distribution.

## 3.3    Illustration of model performance

Figure 1 illustrates the performance of the model on a one dimensional sequence ($D=1$) generated from the changepoint model with $T=160$, $\alpha=0.05$, and $K=10$. The Gibbs sampler was run for $M=30$ iterations and S=200 samples were collected. The top panel shows the actual changepoints (triangles) and the distribution of changepoint assignments averaged over samples. The bottom panel shows the observed data $y$ (thin lines) as well as the $\theta$ values in the generative model (rescaled between 0 and 10).

At locations with large changes between observations, the marginal changepoint probability is quite high. At other locations, the true change in the mean is very small, and the model is less likely to put in a changepoint. The lower right panel shows the distribution over predicted $\theta_{t+1}$ values.

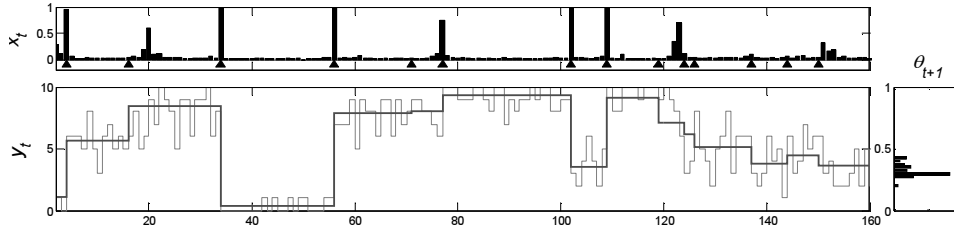

Figure 1. Results of model simulation.

## 4    Prediction experiment

We tested performance of 9 human observers in the prediction task. The observers included the authors, a visitor, and one student who were aware of the statistical nature of the task as well as naïve students. The observers were seated in front of an LCD touch screen displaying a two-dimensional grid of 11 x 11 buttons. The changepoint model was used to generate a sequence of $T=1500$ stimuli for two binomial variables $y_1$ and $y_2$ ($D=2$, $K=10$). The change probability $\alpha$ was set to 0.1. The two variables $y_1$ and $y_2$ specified the two-dimensional button location. The same sequence was used for all observers.

On each trial, the observer touched a button on the grid displayed on the touch screen. Following each button press, the button corresponding to the next $\{y_1,y_2\}$ outcome in the sequence was highlighted. Observers were instructed to press the button that best predicted the next location of the highlighted button. The 1500 trials were divided into

three blocks of 500 trials. Breaks were allowed between blocks. The whole experiment lasted between 15 and 30 minutes. Figure 2 shows the first 50 trials from the third block of the experiment. The top and bottom panels show the actual outcomes for the $y_1$ and $y_2$ button grid coordinates as well as the predictions for two observers (SB and MY). The figure shows that at trial 15, the $y_1$ and $y_2$ coordinates show a large shift followed by an immediate shift in observer's MY predictions (on trial 16). Observer SB waits until trial 17 to make a shift.

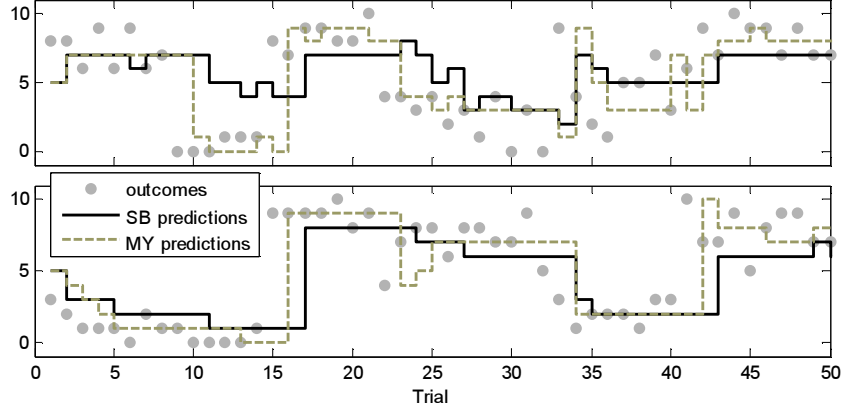

Figure 2. Trial by trial predictions from two observers.

## 4.1  Task error

We assessed prediction performance by comparing the prediction with the actual outcome in the sequence. Task error was measured by normalized city-block distance

$$\text{task error} = \frac{1}{(T-1)} \sum_{t=2}^{T} \left| y_{t,1} - y_{t,1}^{O} \right| + \left| y_{t,2} - y_{t,2}^{O} \right| \tag{4}$$

where $y^{O}$ represents the observer's prediction. Note that the very first trial is excluded from this calculation. Even though more suitable probabilistic measures for prediction error could have been adopted, we wanted to allow comparison of observer's performance with both probabilistic and non-probabilistic models. Task error ranged from 2.8 (for participant MY) to 3.3 (for ML). We also assessed the performance of five models – their task errors ranged from 2.78 to 3.20. The Bayesian models (Section 3) had the lowest task errors, just below 2.8. This fits with our definition of the Bayesian models as "ideal observer" models – their task error is lower than any other model's and any human observer's task error. The fast and frugal models (Section 5) had task errors ranging from 2.85 to 3.20.

## 5  Modeling Results

We will refer to the models with the following letter codes: B=Bayesian Model, LB=limited Bayesian model, FF1..3=fast and frugal models 1..3. We assessed model fit by comparing the model's prediction against the human observers' predictions, again using a normalized city-block distance

$$\text{model error} = \frac{1}{(T-1)} \sum_{t=2}^{T} \left| y_{t,1}^{M} - y_{t,1}^{O} \right| + \left| y_{t,2}^{M} - y_{t,2}^{O} \right| \tag{5}$$

where $y^{M}$ represents the model's prediction. The model error for each individual observer is shown in Figure 3. It is important to note that because each model is associated with a set of free parameters, the parameters optimized for task error and model error are different. For Figure 3, the parameters were optimized to minimize Equation (5) for each individual observer, showing the extent to which these models can capture the performance of individual observers, not necessarily providing the best task performance.

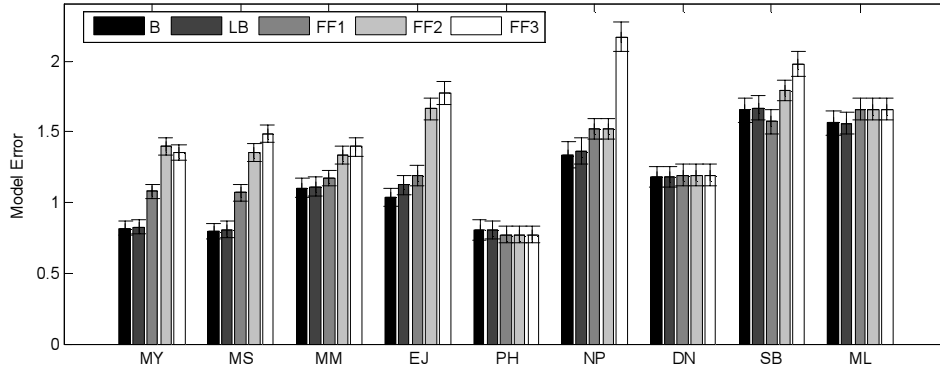

Figure 3. Model error for each individual observer.[1]

## 5.1  Bayesian prediction models

At each trial $t$, the model was provided with the sequence of all previous outcomes. The Gibbs sampling and inference procedures from Eq. (2) and (3) were applied with $M$=30 iterations and $S$=200 samples. The change probability $\alpha$ was a free parameter. In the full Bayesian model, the whole sequence of observations up to the current trial is available for prediction, leading to a memory requirement of up to $T$=1500 trials – a psychologically unreasonable assumption. We therefore also simulated a limited Bayesian model (LB) where the observed sequence was truncated to the last 10 outcomes. The LB model showed almost no decrement in task performance compared to the full Bayesian model. Figure 3 also shows that it fit human data quite well.

## 5.2  Individual Differences

The right-hand panel of Figure 4 plots each observer's task error as a function of the mean city-block distance between their subsequent button presses. This shows a clear U-shaped function. Observers with very variable predictions (e.g., ML and DN) had large average changes between successive button pushes, and also had large task error: These observers were too "twitchy". Observers with very small average button changes (e.g., SB and NP) were not twitchy enough, and also had large task error. Observers in the middle had the lowest task error (e.g., MS and MY). The left-hand panel of Figure 4 shows the same data, but with the x-axis based on the Bayesian model fits. Instead of using mean button change distance to index twitchiness (as in

the right-hand panel), the left-hand panel uses the estimated $\alpha$ parameters from the Bayesian model. A similar U-shaped pattern is observed: individuals with too large or too small $\alpha$ estimates have large task errors.

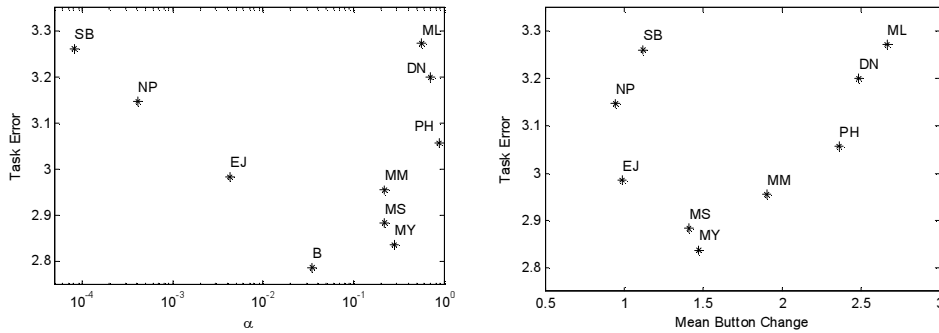

Figure 4. Task error vs. "twitchiness". Left-hand panel indexes twitchiness using estimated $\alpha$ parameters from Bayesian model fits. Right-hand panel uses mean distance between successive predictions.

## 5.3  Fast-and-Frugal (FF) prediction models

These models perform the prediction task using simple heuristics that are cognitively plausible. The FF models keep a short memory of previous stimulus values and make predictions using the same two-step process as the Bayesian model. First, a decision is made as to whether the latent parameter $\theta$ has changed. Second, remembered stimulus values that occurred after the most recently detected changepoint are used to generate the next prediction.

A simple heuristic is used to detect changepoints: If the distance between the most recent observation and prediction is greater than some threshold amount, a change is inferred. We defined the distance between a prediction ($p$) and an observation ($y$) as the difference between the log-likelihoods of $y$ assuming $\theta=p$ and $\theta=y$. Thus, if $f_B(.|\theta, K)$ is the binomial density with parameters $\theta$ and $K$, the distance between observation $y$ and prediction $p$ is defined as $d(y,p)=\log(f_B(y|y,K))-\log(f_B(y|p,K))$. A changepoint on time step $t+1$ is inferred whenever $d(y_t,p_t)>C$. The parameter $C$ governs the twitchiness of the model predictions. If $C$ is large, only very dramatic changepoints will be detected, and the model will be too conservative. If $C$ is small, the model will be too twitchy, and will detect changepoints on the basis of small random fluctuations.

Predictions are based on the most recent $M$ observations, which are kept in memory, unless a changepoint has been detected in which case only those observations occurring after the changepoint are used for prediction. The prediction for time step $t+1$ is simply the mean of these observations, say $p$. Human observers were reticent to make predictions very close to the boundaries. This was modeled by allowing the FF model to change its prediction for the next time step, $y_{t+1}$, towards the mean prediction (0.5). This change reflects a two-way bet. If the probability of a change occurring is $\alpha$, the best guess will be 0.5 if that change occurs, or the mean $p$ if the change does not occur. Thus, the prediction made is actually $y_{t+1}=1/2 \ \alpha+(1-\alpha)p$. Note that we do not allow perfect knowledge of the probability of a changepoint, $\alpha$. Instead, an estimated value of $\alpha$ is used based on the number of changepoints detected in the data series up to time $t$.

The FF model nests two simpler FF models that are psychologically interesting. If the twitchiness threshold parameter $C$ becomes arbitrarily large, the model *never* detects a change and instead becomes a continuous running average model. Predictions from this model are simply a boxcar smooth of the data. Alternatively, if we assume no memory the model must based each prediction on only the previous stimulus (i.e., $M=1$). Above, in Figure 3, we labeled the complete FF model as FF1, the boxcar model as FF2 and the memoryless model FF3.

Figure 3 showed that the complete FF model (FF1) fit the data from all observers significantly better than either the boxcar model (FF2) or the memoryless model (FF3). Exceptions were observers PH, DN and ML, for whom all three FF model fit equally well. This result suggests that our observers were (mostly) doing more than just keeping a running average of the data, or using only the most recent observation. The FF1 model fit the data about as well as the Bayesian models for all observers except MY and MS. Note that, in general, the FF1 and Bayesian model fits are very good: the average city block distance between the human data and the model prediction is around 0.75 (out of 10) buttons on both the x- and y-axes.

# 6   Conclusion

We used an online prediction task to study changepoint detection. Human observers had to predict the next observation in stochastic sequences containing random changepoints. We showed that some observers are too "twitchy": They perform poorly on the prediction task because they see changes where only random fluctuation exists. Other observers are not twitchy enough, and they perform poorly because they fail to see small changes. We developed a Bayesian changepoint detection model that performed the task optimally, and also provided a good fit to human data when sub-optimal parameter settings were used. Finally, we developed a fast-and-frugal model that showed how participants may be able to perform well at the task using minimal information and simple decision heuristics.

## Acknowledgments

We thank Eric-Jan Wagenmakers and Mike Yi for useful discussions related to this work. This work was supported in part by a grant from the US Air Force Office of Scientific Research (AFOSR grant number FA9550-04-1-0317).

## Footnotes

[1] Error bars indicate bootstrapped 95% confidence intervals.

## References

[1] Gilovich, T., Vallone, R. and Tversky, A. (1985). The hot hand in basketball: on the misperception of random sequences. *Cognitive Psychology17*, 295-314.

[2] Albright, S.C. (1993a). A statistical analysis of hitting streaks in baseball. *Journal of the American Statistical Association, 88*, 1175-1183.

[3] Stephens, D.A. (1994). Bayesian retrospective multiple changepoint identification. *Applied Statistics* **43**(1), 159-178.

[4] Carlin, B.P., Gelfand, A.E., & Smith, A.F.M. (1992). Hierarchical Bayesian analysis of changepoint problems. *Applied Statistics* **41**(2), 389-405.

[5] Green, P.J. (1995). Reversible jump Markov chain Monte Carlo computation and Bayesian model determination. *Biometrika* **82**(4), 711-732.

[6] Gigerenzer, G., & Goldstein, D.G. (1996). Reasoning the fast and frugal way: Models of bounded rationality. *Psychological Review, 103*, 650-669.
